# A provably efficient simplex algorithm for classification

**Elad Hazan** *
Technion - Israel Inst. of Tech.
Haifa, 32000
ehazan@ie.technion.ac.il

**Zohar Karnin**
Yahoo! Research
Haifa
zkarnin@ymail.com

## Abstract

We present a simplex algorithm for linear programming in a linear classification formulation. The paramount complexity parameter in linear classification problems is called the *margin*. We prove that for margin values of practical interest our simplex variant performs a polylogarithmic number of pivot steps in the worst case, and its overall running time is near linear. This is in contrast to general linear programming, for which no sub-polynomial pivot rule is known.

## 1 Introduction

Linear programming is a fundamental mathematical model with numerous applications in both combinatorial and continuous optimization. The simplex algorithm for linear programming is a cornerstone of operations research. Despite being one of the most useful algorithms ever designed, not much is known about its theoretical properties.

As of today, it is unknown whether a variant of the simplex algorithm (defined by a pivot rule) exists which makes it run in strongly polynomial time. Further, the simplex algorithm, being a geometrical algorithm that is applied to polytopes defined by linear programs, relates to deep questions in geometry. Perhaps the most famous of which is the "polynomial Hirsh conjecture", that states that the diameter of a polytope is polynomial in its dimension and the number of its facets.

In this paper we analyze a simplex-based algorithm which is guaranteed to run in worst-case polynomial time for large class of practically-interesting linear programs that arise in machine learning, namely linear classification problems. Further, our simplex algorithm performs only a polylogarithmic number of pivot steps and overall *near linear* running time. The only previously known poly-time simplex algorithm performs a polynomial number of pivot steps [KS06].

### 1.1 Related work

The simplex algorithm for linear programming was invented by Danzig [Dan51]. In the sixty years that have passed, numerous attempts have been made to devise a polynomial time simplex algorithm. Various authors have proved polynomial bounds on the number of pivot steps required by simplex variants for inputs that are generated by various distributions, see e.g. [Meg86] as well as articles referenced therein. However, worst case bounds have eluded researchers for many years.

A breakthrough in the theoretical analysis of the simplex algorithm was obtained by Spielman and Teng [ST04], who have shown that its *smoothed complexity* is polynomial, i.e. that the expected running time under a polynomially small perturbation of an arbitrary instance is polynomial. Kelner and Spielman [KS06] have used similar techniques to provide for a worst-case polynomial time simplex algorithm.

In this paper we take another step at explaining the success of the simplex algorithm - we show that for one of the most important and widely used classes of linear programs a simplex algorithm runs in *near linear* time.

We note that more efficient algorithms for linear classification exist, e.g. the optimal algorithm of [CHW10]. The purpose of this paper is to expand our understanding of the simplex method, rather than obtain a more efficient algorithm for classification.

## 2 Preliminaries

### 2.1 Linear classification

Linear classification is a fundamental primitive of machine learning, and is ubiquitous in applications. Formally, we are given a set of vectors-labels pairs $\{\mathbf{A}_i, y_i | i \in [n]\}$, such that $\mathbf{A}_i \in \mathbb{R}^d, y_i \in \{-1, +1\}$ has $\ell_2$ (Euclidean) norm at most one. The goal is to find a hyperplane $\mathbf{x} \in \mathbb{R}^d$ that partitions the vectors into two disjoint subsets according to their sign, i.e. $\text{sign}(\mathbf{A}_i \mathbf{x}) = y_i$. W.l.o.g we can assume that all labels are positive by negating the corresponding vectors of negative labels, i.e. $\forall_i y_i = 1$.

Linear classification can be written as a linear program as follows:

$$\text{find } \mathbf{x} \in \mathbb{R}^d \text{ s.t. } \forall i \in [n] \ \langle \mathbf{A}_i, \mathbf{x} \rangle > 0 \tag{1}$$

The original linear classification problem is then *separable*, i.e. there exists a separating hyperplane, if and only if the above linear program has a feasible solution. Further, any linear program in standard form can be written in linear classification form (1) by elementary manipulations and addition of a single variable (see [DV08] for more details).

Henceforth we refer to a linear program in format (1) by its coefficient matrix $\mathbf{A}$. All vectors are column vectors, and we denote inner products by $\langle \mathbf{x}, \mathbf{y} \rangle$. A parameter of paramount importance to linear classification is the margin, defined as follows

**Definition 1.** *The margin of a linear program in format* (1) *, such that* $\forall_i \|\mathbf{A}_i\| \leq 1$, *is defined as*

$$\lambda = \lambda(\mathbf{A}) = \max_{\|\mathbf{x}\| \leq 1} \min_{i \in [n]} \langle \mathbf{A}_i, \mathbf{x} \rangle$$

*We say that the instance* $\mathbf{A}$ *is a* $\lambda$-*margin LP.*

Notice that we have restricted $\mathbf{x}$ as well as the rows of $\mathbf{A}$ to have bounded norm, since otherwise the margin is ill-defined as it can change by scaling of $\mathbf{x}$. Intuitively, the larger the margin, the easier the linear program is to solve.

While any linear program can be converted to an equivalent one in form (1), the margin can be exponentially small in the representation. However, in practical applications the margin is usually a constant independent of the problem dimensions; a justification is given next. Therefore we henceforth treat the margin as a separate parameter of the linear program, and devise efficient algorithms for solving it when the margin is a constant independent of the problem dimensions.

**Support vector machines - why is the margin large ?** In real-world problems the data is seldom separable. This is due to many reasons, most prominently noise and modeling errors.

Hence practitioners settle for approximate linear classifiers. Finding a linear classifier that minimizes the number of classification errors is NP-hard, and inapproximable [FGKP06]. The relaxation of choice is to minimize the sum of errors, called "soft-margin SVM" (Support Vector Machine) [CV95], and is one of the most widely used algorithms in machine learning. Formally, a soft-margin SVM instance is given by the following mathematical program:

$$\min \sum_i \xi_i$$
$$\forall i \in [n] \ y_i(\langle \mathbf{x}, \mathbf{A}_i \rangle + b) + \xi_i \geq 0 \tag{2}$$
$$\|\mathbf{x}\| \leq 1$$

The norm constraint on $\mathbf{x}$ is usually taken to be the Euclidean norm, but other norms are also common such as the $\ell_1$ or $\ell_\infty$ constraints that give rise to linear programs.

In this paper we discuss the separable case (formulation (1)) alone. The non-separable case turns out to be much easier when we allow an additive loss of a small constant to the margin. We elaborate on this point in Section 6.1. We will restrict our attention only to the case where the bounding norm of $x$ is the $\ell_2$ norm as it is the most common case.

## 2.2 Linear Programming and Smoothed analysis

Smoothed analysis was introduced in [ST04] to explain the excellent performance of the simplex algorithm in practice. A $\sigma$-smooth LP is an LP where each coefficient is perturbed by a Gaussian noise of variance $\sigma^2$.

In their seminal paper, Spielman and Teng proved the existence of a simplex algorithm that solves $\sigma$-smooth LP in polynomial time (polynomial also in $\sigma^{-1}$). Consequently, Vershynin [Ver09] presented a simpler algorithm and significantly improved the running time. In the next sections we will compare our results to the mentioned papers and point out a crucial lemma used in both papers that will also be used here.

## 2.3 Statement of our results

For a separable SVM instance of $n$ variables in a space of $d$ dimensions and margin $\lambda$, we provide a simplex algorithm with at most $\text{poly}(\log(n), \lambda^{-1})$ many pivot steps. Our statement is given for the $\ell_2$-SVM case, that is the case where the vector $w$ (see Definition 1) has a bounded $\ell_2$ norm. The algorithm achieves a solution with margin $O(\sqrt{\log(n)/d})$ when viewed as a separator in the $d$ dimensional space. However, in an alternative yet (practically) equivalent view, the margin of the solution is in fact arbitrarily close to $\lambda$.

**Theorem 1.** *Let $\mathcal{L}$ be a separable $\ell_2$-SVM instance of dimension $d$ with $n$ examples and margin $\lambda$. Assume that $\lambda > c_1\sqrt{\log n / d}$ where $c_1$ is some sufficiently large universal constant. Let $0 < \varepsilon < \lambda$ be a parameter. The simplex algorithm presented in this paper requires $\tilde{O}(nd)$ preprocessing time and $\text{poly}(\varepsilon^{-1}, \log(n))$ pivot steps. The algorithm outputs a subspace $V \subseteq \mathbb{R}^d$ of dimension $k = \Theta(\log(n)/\varepsilon^2)$ and a hyperplane within it. The margin of the solution when viewed as a hyperplane in $\mathbb{R}^d$ is $O(\sqrt{\log(n)/d})$. When projecting the data points onto $V$, the margin of the solution is $\lambda - \varepsilon$.*

In words, the above theorem states that when viewed as a classification problem the obtained margin is almost optimal. We note that when classifying a new point one does not have to project it to the subspace $V$, but rather assign a sign according to the classifying hyperplane in $\mathbb{R}^d$.

**Tightness of the Generalization Bound**    In first sight it seems that our result gives a week generalization bound since the margin obtained in the original dimension is low. However, the margin of the found solution in the *reduced dimension* (i.e., within $V$) is almost optimal (i.e. $\lambda - \varepsilon$ where $\lambda$ is the optimal margin). It follows that the generalization bound essentially the same one obtained by an exact solution.

**LP perspective and the smoothed analysis framework**    As mentioned earlier, any linear program can be viewed as a classification LP by introducing a single new variable. Furthermore, any solution with a positive margin translates into an optimal solution to the original LP. Our algorithm solves the classification LP in a sub-optimal manner in the sense that it does not find a separator with an optimal margin. However, in the perspective of a general LP solver[1], the solution is optimal as any positive margin suffices. It stands to reason that in many practical settings the margin of the solution is constant or polylogarithmically small at worst. In such cases, our simplex algorithm solves the LP by using at most a polylogarithmic number of pivot steps. We further mention that without the large margin assumption, in the smoothed analysis framework it is known ([BD02], Lemma 6.2) that the margin is w.h.p. polynomially bounded by the parameters. Hence, our algorithm runs in polynomial time in the smoothed analysis framework as well.

## 3  Our Techniques

The process involves five preliminary steps. Reducing the dimension, adding artificial constraints to bound the norm of the solution, perturbing the low dimensional LP, finding a feasible point and shifting the polytope. The process of reducing the dimension is standard. We use the Johnson and Lindenstrauss Lemma [JL84] to reduce the dimension of the data points from $d$ to $k = O(\log(n)/\varepsilon^2)$ where $\varepsilon$ is an error parameter that can be considered as a constant. This step reduces the time complexity by reducing both the number and running time of the pivot steps. In order to bound the $\ell_2$ norm of the original vector, we bound the $\ell_\infty$ norm of the low dimensional vector. This will eventually result in a multiplicative loss of $\sqrt{\log(k)}$ to the margin. We note that we could have avoided this loss by bounding the $\ell_1$ norm of the vector at the cost of a more technically involved proof. Specifically, one should bound the $\ell_1$ norm of the embedding of the vector into a space where the $\ell_1$ and $\ell_2$ norms behave similarly, up to a multiplicative distortion of $1 \pm \varepsilon$. Such an embedding of $\ell_2^k$ in $\ell_1^K$ exists for $K = O(k/\varepsilon^2)$ [Ind00]. Another side effect is a larger constant in the polynomial dependence of $\varepsilon$ in the running time.

The perturbation step involves adding a random Gaussian noise vector to the matrix of constraints, where the amplitude of each row is determined by the norm of the corresponding constraint vector. This step ensures the bound on the number of pivot step performed by the simplex algorithm. In order to find a feasible point we exploit the fact that when the margin is allowed to be negative, there is always a feasible solution. We prove for a fixed set of constraints, one of which is a negative lower bound on the margin, that the corresponding point $\mathbf{v}_0$ is not only feasible but is the unique optimal solution for fixed direction. The direction is independent of the added noise, which is a necessary property when bounding the number of pivot steps.

Our final step is a shift of the polytope. Since we use the shadow vertex pivot rule we must have an LP instance for which $\mathbf{0}$ is an interior point of the polytope. This property is not held for our polytope as the LP contains inequalities of the form $\langle \mathbf{a}, \mathbf{x} \rangle \geq 0$. However, we prove that both $\mathbf{0}$ and $\mathbf{v}_0$ are feasible solution to the LP that do not share a common facet. Hence, their average is an interior point of the polytope and a shift by $-\mathbf{v}_0/2$ would ensure that $\mathbf{0}$ is an interior point as required.

Once the preprocessing is done we solve the LP via the shadow vertex method which is guaranteed to finish after a polylogarithmic number of pivot steps. Given a sufficiently small additive noise and sufficiently large target dimension we are guaranteed that the obtained solution is an almost optimal solution to the unperturbed low dimensional problem and a $\tilde{O}(\sqrt{k/d})$ approximation to the higher dimensional problem.

## 4  Tool Set

### 4.1  Dimension reduction

The Johnson-Lindenstrauss Lemma [JL84] asserts that one can project vectors onto a lower dimensional space and roughly preserve their norms, pairwise distances and inner products. The following is an immediate consequence of Theorem 2.1 and Lemma 2.2 of [DG03].

**Theorem 2.** *Let $\varepsilon$ 0 and let $k, d$ be integers where $d > k > 9/\varepsilon^2$. Consider a linear projection $M : \mathbb{R}^d \mapsto \mathbb{R}^k$ onto a uniformly chosen subspace[2]. For any pair of fixed vector $\mathbf{u}, \mathbf{v} \in \mathbb{R}^d$ where $\|\mathbf{u}\|, \|\mathbf{v}\| \leq 1$, it holds that*

$$\Pr\left[\left|\|\mathbf{u}\|^2 - \|M\mathbf{u}\|^2\right| > \varepsilon\right] < \exp(-k\varepsilon^2/9)$$
$$\Pr\left[|\langle \mathbf{u}, \mathbf{v} \rangle - \langle M\mathbf{u}, M\mathbf{v} \rangle| > 3\varepsilon\right] < 3\exp(-k\varepsilon^2/9)$$

### 4.2  The number of vertices in the shadow of a perturbed polytope

A key lemma in the papers of [ST04, Ver09] is a bound on the expected number of vertices in the projection of a perturbed polytope onto a plane. The following geometric theorem is will be used in our paper:

**Theorem 3** ([Ver09] Theorem 6.2). *Let $\mathbf{A}_1, ..., \mathbf{A}_n$ be independent Gaussian vectors in $\mathbb{R}^d$ with centers of norm at most 1, and whose varaince satisfies:*

$$\sigma^2 \leq \frac{1}{36d \log n}$$

*Let $E$ be a fixed plane in $\mathbb{R}^d$. Then the random polytope $P = \text{conv}(0, \mathbf{A}_1, ..., \mathbf{A}_n)$ satisfies*

$$\mathbf{E}[|\text{edges}(P \cap E)|] = O(d^3 \sigma^{-4})$$

### 4.3  The shadow vertex method

The shadow vertex method is a pivot rule used to solve LPs. In order to apply it, the polytope of the LP must have $\mathbf{0}$ as an interior point. Algebraically, all the inequalities must be of the form $\langle \mathbf{a}, \mathbf{x} \rangle \leq 1$ (or alternatively $\langle \mathbf{a}, \mathbf{x} \rangle \leq b$ where $b > 0$). The input consists of a feasible point $\mathbf{v}$ in the polytope and a direction $\mathbf{u}$ in which it is farthest, compared to all other feasible points. In a nutshell, the method involves gradually turning the vector $\mathbf{u}$ towards the direction of the target direction $\mathbf{c}$, while traversing through the optimal solutions to the temporary direction at every stage. For more on the shadow vertex method we refer the reader to [ST04], Section 3.2

The manner in which Theorem 3 is used, both in the above mentioned papers and the current one, is the following. Consider an LP of the form

$$\max \mathbf{c}^\top \mathbf{x}$$
$$\forall i \in [n] \quad \langle \mathbf{A}_i, \mathbf{x} \rangle \leq 1$$

When solving the LP via the shadow vertex method, the number of pivot steps is upper bounded by the number of edges in $P \cap E$ where $P = \text{conv}(0, \mathbf{A}_1, ..., \mathbf{A}_n)$ and $E$ is the plane spanned by the target direction $\mathbf{c}$ and the initial direction $\mathbf{u}$ obtained in the phase-1 step.

## 5  Algorithm and Analysis

Our simplex variant is defined in Algorithm 1 below. It is composed of projecting the polytope onto a lower dimension, adding noise, finding an initial vertex (Phase 1), shifting and applying the shadow vertex simplex algorithm [GS55].

**Theorem 4.** *Algorithm 1 performs an expected number of $O(\text{poly}(\log n, \frac{1}{\lambda}))$ pivot steps. Over instance $\mathbf{A}$ with $\lambda$-margin it returns, with probability at least $1 - O(\frac{1}{k} + \frac{1}{n})$, a feasible solution $\bar{\mathbf{x}}$ with margin $\Omega(\frac{\lambda \sqrt{k}}{\sqrt{d \log k}})$.*

Note that the algorithm requires knowledge of $\lambda$. This can be overcome with a simple binary search. To prove Theorem 4, we first prove several auxilary lemmas. Due to space restrictions, some of the proofs are replaced with a brief sketch.

**Lemma 5.** *With probability at least $1 - 1/k$ there exists a feasible solution to $LP_{\text{bounded}}$, denoted $(\hat{\mathbf{x}}, \tau)$ that satisfies $\tau \geq \lambda - \varepsilon$ and $\|\hat{\mathbf{x}}\|_\infty \leq 5 \frac{\sqrt{\log(k)}}{\sqrt{k}}$.*

*Proof Sketch.* Since $\mathbf{A}$ has margin $\lambda$, there exists $\mathbf{x}^* \in \mathbb{R}^d$ such that $\forall i . \langle \mathbf{A}_i, \mathbf{x}^* \rangle \geq \lambda$ and $\|\mathbf{x}^*\|_2 = 1$. We use Theorem 2 to show that the projection of $x^*$ has, w.h.p., both a large margin and a small $\ell_\infty$ norm.

$\square$

Denote the $k + 1$ dimensional noise vectors that were added in step 3 by $\mathbf{err}_1, \ldots, \mathbf{err}_{n+2k}$. The following lemma will provide some basic facts that occur w.h.p. for the noise vectors. The proof is an easy consequence of the 2-stability of Gaussians, and standard tail bounds of the Chi-Squared distribution and is thus omitted.

**Lemma 6.** *Let $\mathbf{err}_1, \ldots, \mathbf{err}_{n+2k}$ be defined as above:*

1. *w.p. at least $1 - 1/n$, $\forall i$, $\|\mathbf{err}_i\|_2 \leq O(\sigma \sqrt{k \log(n)}) \leq \frac{1}{20\sqrt{k}}$*

---

**Algorithm 1** large margin simplex

---
1: Input: a $\lambda$-margin LP instance $\mathbf{A}$.
2: Let $\varepsilon = \frac{\lambda}{6}, k = \frac{9 \cdot 16^2 \log(n/\varepsilon)}{\varepsilon^2}, \sigma^2 = \frac{1}{100 k \log k \log n}$.
3: (step 1: dimension reduction) Generate $\mathbf{M} \in \mathbb{R}^{k \times d}$, a projection onto a random $k$-dimensional subspace. Let $\hat{\mathbf{A}} \in \mathbb{R}^{n \times (k+1)}$ be given by $\hat{\mathbf{A}}_i = (\sqrt{\frac{d}{k}} \mathbf{M} \mathbf{A}_i, -1)$
4: (step 2: bounding $\|\mathbf{x}\|$) Add the $k$ constraints $\langle \mathbf{e}_i, \mathbf{x} \rangle \geq -\frac{6\sqrt{\log k}}{\sqrt{k}}$, the $k$ constraints $\langle -\mathbf{e}_i, \mathbf{x} \rangle \geq -\frac{6\sqrt{\log k}}{\sqrt{k}}$, and one additional constraint $\tau \geq -8\sqrt{\log(k)}$. Denote the coefficient vectors $\hat{\mathbf{A}}_{n+1}, \ldots, \hat{\mathbf{A}}_{n+2k}$ and $\hat{\mathbf{A}}_0$ correspondingly. We obtain the following LP denoted by $LP_{\text{bounded}}$,

$$\max \langle \mathbf{e}_{k+1}, (\mathbf{x}, \tau) \rangle \tag{3}$$
$$\forall i \in [0, \ldots, n+2k] \,.\, \left\langle \hat{\mathbf{A}}_i, (\mathbf{x}, \tau) \right\rangle \geq b_i$$

5: (step 3: adding noise) Add a random independently distributed Gaussian noise to every entry of every constraint vector except $\hat{\mathbf{A}}_0$ according to $\mathcal{N}(0, \sigma^2 \cdot \|\hat{\mathbf{A}}_i\|_2^2)$. Denote the resulting constraint vectors as $\tilde{\mathbf{A}}_i$. Denote the resulting LP by $LP_{\text{noise}}$.
6: (step 4: phase-1): Let $\mathbf{v}_0 \in \mathbb{R}^{k+1}$ be the vertex for which inequalities $0, n+k+1, \ldots, n+2k$ are held as equalities. Define $\mathbf{u}_0 \in \mathbb{R}^{k+1}$ as $\mathbf{u}_0 \triangleq (1, \ldots, 1, -1)$.
7: (step 5: shifting the polytope) For all $i \in [0, \ldots, n+2k]$, change the value of $b_i$ to $\hat{b}_i \triangleq b_i + \left\langle \tilde{\mathbf{A}}_i, \mathbf{v}_0/2 \right\rangle$.
8: (step 6 - shadow vertex simplex): Let $E = \text{span}(\mathbf{u}_0, \mathbf{e}_{k+1})$. Apply the shadow vertex simplex algorithm on the polygon which is the projection of $\text{conv}\{\mathcal{V}\}$, where $\mathcal{V} = \{0, \hat{\mathbf{A}}_0/\hat{b}_0, \tilde{\mathbf{A}}_1/\hat{b}_1, \ldots, \tilde{\mathbf{A}}_{n+2k}/\hat{b}_{n+2k}\}$, onto $E$. Let the solution be $\tilde{\mathbf{x}}$.
9: **return** $\frac{\bar{\mathbf{x}}}{\|\bar{\mathbf{x}}\|_2}$ where $\bar{\mathbf{x}} = \mathbf{M}^\top (\tilde{\mathbf{x}} + \mathbf{v}_0/2)$

---

2. Fix some $I \subset [n+2k]$ of size $|I| = k$ and define $B_I$ be the $(k+1) \times (k+1)$ matrix whose first $k$ columns consist of $\{\mathbf{err}_i\}_{i \in I}$ and $k+1$ column is the 0 vector. W.p. at least $1 - 1/n$ it holds that the top singular value of $B_I$ is at most $1/2$. Furthermore, w.p. at least $1 - 1/n$ the 2-norms of the rows of $B$ are upper bounded by $\frac{1}{4\sqrt{k+1}}$.

**Lemma 7.** *Let* $\hat{\mathbf{A}}, \tilde{\mathbf{A}}, \hat{\mathbf{x}} \in \mathbb{R}^k$ *be as above. Then with probability at least* $1 - O(1/k)$:

1. *for* $\tau = \lambda - 2\varepsilon$, *the point* $(\hat{\mathbf{x}}, \tau) \in \mathbb{R}^{k+1}$ *is a feasible solution of* $LP_{\text{noise}}$.

2. *for every* $x \in \mathbb{R}^k$, $\tau \in \mathbb{R}$ *where* $(\mathbf{x}, \tau)$ *is a feasible solution of the* $LP_{\text{noise}}$ *it holds that*

$$\|\mathbf{x}\|_\infty \leq 7\frac{\sqrt{\log(k)}}{\sqrt{k}}, \qquad \forall i \,.\, \left| \left\langle \tilde{\mathbf{A}}_i, (\mathbf{x}, \tau) \right\rangle - \left\langle \hat{\mathbf{A}}_i, (\mathbf{x}, \tau) \right\rangle \right| \leq \frac{\sqrt{\log k}}{\sqrt{k}}$$

Proof of this Lemma is deferred to the full version of this paper.

**Lemma 8.** *with probability* $1 - O(1/k)$, *the vector* $\mathbf{v}_0$ *is a basic feasible solution (vertex) of* $LP_{\text{noise}}$

*proof sketch.* The vector $\mathbf{v}_0$ is a basic solution as it is defined by $k+1$ equalities. To prove that is feasible we exploit the fact that the last entry corresponding to $\tau$ is sufficiently small and that all of the constraints are of the form $\langle a, x \rangle \geq \tau$. $\qquad \square$

The next lemma provides us with a direction $\mathbf{u}_0$ for which $\mathbf{v}_0$ is the unique optimal solution w.r.t to the objective $\max_{\mathbf{x} \in \mathcal{P}} \langle \mathbf{u}_0, \mathbf{x} \rangle$, where $\mathcal{P}$ is the polytope of $LP_{\text{noise}}$. The vector $\mathbf{u}_0$ is independent of the added noise. This is crucial for the following steps.

**Lemma 9.** *Let* $\mathbf{u}_0 = (1, \ldots, 1, -1)$. *With probability at least* $1 - O(1/n)$, *the point* $\mathbf{v}_0$ *is the optimal solution w.r.t to the objective* $\max_{\mathbf{x} \in \mathcal{P}} \langle \mathbf{u}_0, \mathbf{x} \rangle$, *where* $\mathcal{P}$ *is the polytope of* $LP_{\text{noise}}$.

*Proof Sketch.* The set of points $\mathbf{u}$ in which $\mathbf{v}_0$ is the optimal solution is defined by a (blunt) cone $\{\sum_i \alpha_i \mathbf{a}_i | \forall i, \ \alpha_i > 0\}$, where $\mathbf{a}_i = -\tilde{\mathbf{A}}_{n+k+i}$ for $i \in [k]$, $\mathbf{a}_{k+1} = -\hat{\mathbf{A}}_0$. Consider the cone corresponding to the constraints $\hat{\mathbf{A}}$; $\mathbf{u}_0$ resides in its interior, far away from its boarders. Specifically, $\mathbf{u}_0 = \sum_{i=1}^{k}(-\hat{\mathbf{A}}_{n+k+i}) + (-\hat{\mathbf{A}}_0)$. Since the difference between $\hat{\mathbf{A}}_i$ and $\tilde{\mathbf{A}}_i$ is small w.h.p., we get that $\mathbf{u}_0$ resides, w.h.p., in the cone of points in which $\mathbf{v}_0$ is optimal, as required. $\qquad\square$

**Lemma 10.** *The point $\mathbf{v}_0/2$ is a feasible interior point of the polytope with probability at least $1 - O(1/n)$.*

*Proof.* By Lemma 9, $\mathbf{v}_0$ is a feasible point. Also, according to its definition it is clear that w.p 1, it lies on $k + 1$ facets of the polytope, neither of which contains the point $\mathbf{0}$. In other words, no facet contains both $\mathbf{v}_0$ and $\mathbf{0}$. Since $\mathbf{0}$ is clearly a feasible point of the polytope, we get that $\mathbf{v}_0/2$ is a feasible interior point as claimed. $\qquad\square$

*Proof of Theorem 4.* We first note that in order to use the shadow vertex method, $\mathbf{0}$ must be an interior point of the polytope. This does not happen in the original polytope, hence the shift of step 5. Indeed according to Lemma 10, $\mathbf{v}_0/2$ is an interior point of the polytope, and by shifting it to $\mathbf{0}$, the shadow vertex method can indeed be implemented.

We will assume that the statements of the auxiliary lemmas are held. This happens with probability at least $1 - O(\frac{1}{k} + \frac{1}{n})$, which is the stated success probability of the algorithm. By Lemma 7, $LP_{\text{noise}}$ has a basic feasible solution with $\tau \geq \lambda - 2\varepsilon$. The vertex $\mathbf{v}_0$, along with the direction $\mathbf{u}_0$ which it optimizes, is a feasible starting vector for the shadow vertex simplex algorithm on the plane $E$, and hence applying the simplex algorithm with the shadow vertex pivot rule will return a basic feasible solution in dimension $k + 1$, denoted $(\tilde{\mathbf{x}}, \tau')$, for which $\forall i \in [n]$ . $\left\langle \tilde{\mathbf{A}}_i, (\tilde{\mathbf{x}}, \tau') \right\rangle \geq 0$ and $\tau' \geq \lambda - 2\varepsilon$. Using Lemma 7 part two, we have that for all $i \in [n]$,

$$\left\langle \hat{\mathbf{A}}_i, (\tilde{\mathbf{x}}, \tau') \right\rangle \geq \left\langle \tilde{\mathbf{A}}_i, (\tilde{\mathbf{x}}, \tau') \right\rangle - \frac{\sqrt{\log k}}{\sqrt{k}} \geq -\varepsilon \ \Rightarrow \ \langle \mathbf{M}\mathbf{A}_i, \tilde{\mathbf{x}} \rangle \geq \lambda - 3\varepsilon. \qquad (4)$$

Since $\bar{\mathbf{x}} = \sqrt{d/k}\mathbf{M}^\top \tilde{\mathbf{x}}$, we get that for all $i \in [n]$, $\langle \mathbf{A}_i, \bar{\mathbf{x}} \rangle = \sqrt{d/k}\mathbf{A}_i^\top \mathbf{M}^\top \tilde{\mathbf{x}} = \langle f(\mathbf{A}_i), \tilde{\mathbf{x}} \rangle \geq \lambda - 3\varepsilon$ and this provides a solution to the original LP.

To compute the margin of this solution, note that the rows of $M$ consist of an orthonormal set. Hence, by Lemma 7, $\|M^\top \tilde{\mathbf{x}}\|_2 = \|\tilde{\mathbf{x}}\|_2 \leq 7\sqrt{\log(k)}$ meaning that $\|\bar{\mathbf{x}}\|_2 \leq 7\sqrt{\log(k)d/k}$. It follows that the margin of the solution is at least $\geq (\lambda - 3\varepsilon) \cdot \sqrt{k}/(7\sqrt{\log(k)d})$

**Running time:** The number of steps in this simplex step is bounded by the number of vertices in the polygon which is the projection of the polytope of $LP_{\text{noise}}$ onto the plane $E = \text{span}\{\mathbf{u}_0, \mathbf{v}_T\}$. Let $\mathcal{V} = \{\tilde{\mathbf{A}}\}_{i=1}^{n+2k}$. Since all of the points in $\mathcal{V}$ are perturbed, the number of vertices in the polygon $\text{conv}(\mathcal{V}) \cap E$ is bounded w.h.p. as in Theorem 3 by $O(k^3\sigma^{-4}) = \tilde{O}(\log^{11}(n)/\lambda^{14})$. Since the points $0, \hat{\mathbf{A}}_0$ reside in the plane $E$, the the number of vertices of $(\text{conv}(\mathcal{V} \cup \{0, \hat{\mathbf{A}}_0\})) \cap E$ is at most the number of vertices in $\text{conv}(\mathcal{V}) \cap E$ plus 4, which is asymptotically the same. Each pivot step in the shadow vertex simplex method can be implemented to run in time $O(nk) = \tilde{O}(n/\lambda^{14})$ for $n$ constraints in dimension $k$. The dimension reduction step required $\tilde{O}(nd)$ time. All other operations including adding noise and shifting the polytope are faster than the shadow vertex simplex procedure, leading to an overall running time of $\tilde{O}(nd)$ (assuming $\lambda$ is a constant or sub polynomial in $d$). $\qquad\square$

*proof of Theorem 1.* The statement regarding the margin of the solution, viewed as a point in $\mathbb{R}^d$ is immediate from Theorem 4. To prove the claim regarding the view in the low dimensional space, consider Equation 4 in the above proof. Put in words, it states the following: Consider the projection $M$ of the algorithm (or alternatively its image $V$) and the classification problem of the points projected onto $V$. The margin of the solution produced by the algorithm (i.e., of $\tilde{\mathbf{x}}$) is at least $\lambda - 3\varepsilon$. The $\ell_\infty$-norm $\tilde{\mathbf{x}}$ of is clearly bounded by $O(\sqrt{\log(k)/k})$. Hence, the margin of the normalized point $\tilde{\mathbf{x}}/\|\tilde{\mathbf{x}}\|_2$ is $\Omega(\lambda/\sqrt{\log(k)})$. In order to achieve a margin of $\lambda - O(\varepsilon)$, one should replace the $\ell_\infty$ bound in the LP with an approximate $\ell_2$ bound. This can be done via linear constraints by bounding

the $\ell_1$ norm of $Fx$ where $F : \mathbb{R}^k \to \mathbb{R}^K$, $K = O(k/\varepsilon^2)$ and $F$ has the property that for every $x \in \mathbb{R}^k$, $\left| \frac{\|Fx\|_1}{\|x\|_2} - 1 \right| < \varepsilon$. A properly scaled matrix of i.i.d. Gaussians has this property [Ind00]. This step would eliminate the need for the extra $\sqrt{\log(k)}$ factor. The other multiplicative constants can be reduced to $1 + O(\varepsilon)$, thus ensuring the norm of $\tilde{\mathbf{x}}$ is at most $1 + O(\varepsilon)$, by assigning a slightly smaller value for $\sigma$; specifically, $\sigma/\varepsilon$ would do. Once the 2-norm of $\tilde{\mathbf{x}}$ is bounded by $1 + O(\varepsilon)$, the margin of the normalized point is $\lambda - O(\varepsilon)$. $\qquad\square$

## 6 Discussion

The simplex algorithm for linear programming is a cornerstone of operations research whose computational complexity remains elusive despite decades of research. In this paper we examine the simplex algorithm in the lens of machine learning, and in particular via linear classification, which is equivalent to linear programming. We show that in the cases where the margin parameter is large, say a small constant, we can construct a simplex algorithm whose worst case complexity is (quasi) linear. Indeed in many practical problems the margin parameter is a constant unrelated to the other parameters. For example, in cases where a constant inherent noise exists, the margin must be large otherwise the problem is simply unsolvable.

### 6.1 soft margin SVM

In the setting of this paper, the case of soft margin SVM turns out to be algorithmically easier to solve than the separable case. In a nutshell, the main hardship in the separable case is that a large number of data points may be problematic. This is since the separating hyperplane must separate *all* of the points and not most of them, meaning that every one of the data points must be taken in consideration. A more formal statement is the following. In our setting we have three parameters. The number of points $n$, the dimension $d$ and the 'sub optimality' $\varepsilon$. In the soft margin (e.g. hinge loss) case, the number of points may be reduced to $\text{poly}(\varepsilon^{-1})$ by elementary methods. Specifically, it is in easy task to prove that if we omit all but a random subset of $\log(\varepsilon^{-1})/\varepsilon^2$ data points, the hinge loss corresponding to the obtained separator w.r.t the full set of points will be $O(\varepsilon)$. In fact, it suffices to solve the problem with the reduced number of points, up to an additive loss of $\varepsilon$ to the margin to obtain the same result. As a consequence of the reduced number of points, the dimension can be reduced, analogously to the separable case to $d' = O(\log(\varepsilon^{-1})/\varepsilon^2)$.

The above essentially states that the original problem can be reduced, by performing a single pass over the input (perhaps even less than that), to one where all the only parameter is $\varepsilon$. From this point, the only challenge is to solve the resulting LP, up to an $\varepsilon$ additive loss to the optimum, in time polynomial to its size. There are many methods available for this problem.

To conclude, the soft margin SVM problem is much easier than the separable case hence we do not analyze it in this paper.

## Footnotes

*Work conducted at and funded by the Technion-Micorsoft Electronic Commerce Research Center

[1]The statement is true only for feasibility LPs. However, any LP can be transformed into a feasibility LP by performing a binary search for its solution value.

[2]Alternatively, $M$ can be viewed as the composition of a random rotation $U$ followed by taking the first $k$ coordinates

## References

[BD02]     A. Blum and J. Dunagan. Smoothed analysis of the perceptron algorithm for linear programming. In *Proceedings of the thirteenth annual ACM-SIAM symposium on Discrete algorithms*, pages 905–914. Society for Industrial and Applied Mathematics, 2002.

[CHW10]  Kenneth L. Clarkson, Elad Hazan, and David P. Woodruff. Sublinear optimization for machine learning. In *FOCS*, pages 449–457. IEEE Computer Society, 2010.

[CV95]     Corinna Cortes and Vladimir Vapnik. Support-vector networks. In *Machine Learning*, pages 273–297, 1995.

[Dan51]    G. B. Dantzig. Maximization of linear function of variables subject to linear inequalities. *Activity Analysis of Production and Allocation*, pages 339–347, 1951.

[DG03]     Sanjoy Dasgupta and Anupam Gupta. An elementary proof of a theorem of johnson and lindenstrauss. *Random Struct. Algorithms*, 22:60–65, January 2003.

[DV08]     John Dunagan and Santosh Vempala. A simple polynomial-time rescaling algorithm for solving linear programs. *Math. Program.*, 114(1):101–114, 2008.

[FGKP06] Vitaly Feldman, Parikshit Gopalan, Subhash Khot, and Ashok Kumar Ponnuswami. New results for learning noisy parities and halfspaces. In *FOCS*, pages 563–574. IEEE Computer Society, 2006.

[GS55] S. Gass and T. Saaty. The computational algorithm for the parameteric objective function. *Naval Research Logistics Quarterly*, 2:39–45, 1955.

[Ind00] P. Indyk. Stable distributions, pseudorandom generators, embeddings and data stream computation. In *Foundations of Computer Science, 2000. Proceedings. 41st Annual Symposium on*, pages 189–197. IEEE, 2000.

[JL84] W. B. Johnson and J. Lindenstrauss. Extensions of lipschitz mapping into hilbert space. *Contemporary Mathematics*, 26:189–206, 1984.

[KS06] Jonathan A. Kelner and Daniel A. Spielman. A randomized polynomial-time simplex algorithm for linear programming. In *Proceedings of the thirty-eighth annual ACM symposium on Theory of computing*, STOC '06, pages 51–60, New York, NY, USA, 2006. ACM.

[Meg86] Nimrod Megiddo. Improved asymptotic analysis of the average number of steps performed by the self-dual simplex algorithm. *Math. Program.*, 35:140–172, June 1986.

[ST04] Daniel A. Spielman and Shang-Hua Teng. Smoothed analysis of algorithms: Why the simplex algorithm usually takes polynomial time. *J. ACM*, 51:385–463, May 2004.

[Ver09] Roman Vershynin. Beyond hirsch conjecture: Walks on random polytopes and smoothed complexity of the simplex method. *SIAM J. Comput.*, 39(2):646–678, 2009.

